# Direct Multi-Step Time Series Prediction Using TD(λ)

**Peter T. Kazlas**
Department of Electrical
and Computer Engineering
University of Colorado
Boulder, CO 80309-0425
pkazlas@colorado.edu

**Andreas S. Weigend**
Department of Computer Science
and Institute of Cognitive Science
University of Colorado
Boulder, CO 80309-0430
andreas@cs.colorado.edu *

## Abstract

This paper explores the application of Temporal Difference (TD) learning (Sutton, 1988) to forecasting the behavior of dynamical systems with real-valued outputs (as opposed to game-like situations). The performance of TD learning in comparison to standard supervised learning depends on the amount of noise present in the data. In this paper, we use a deterministic chaotic time series from a low-noise laser. For the task of direct five-step ahead predictions, our experiments show that standard supervised learning is better than TD learning. The TD algorithm can be viewed as linking adjacent predictions. A similar effect can be obtained by sharing the internal representation in the network. We thus compare two architectures for both paradigms: the first architecture ("separate hidden units") consists of individual networks for each of the five direct multi-step prediction tasks; the second ("shared hidden units") has a single (larger) hidden layer that finds a representation from which all five predictions for the next five steps are generated. For this data set we do not find any significant difference between the two architectures.

*http://www.cs.colorado.edu/~andreas/Home.html.
This paper is available with figures in colors as ftp://ftp.cs.colorado.edu/pub/Time-Series/MyPapers/kazlas.weigend_nips7.ps.Z .

# 1  Introduction

The *Santa Fe Time Series Prediction and Analysis Competition* (Weigend & Gershenfeld, 1994) saw a relatively large number of different nonlinear techniques applied to the prediction of a few time series. One of the results was that some neural networks did very well (but incidentally some other neural networks also did very poorly). All neural networks were trained with standard supervised learning where the network is trained based on differences between the predicted and observed values of the series. The differences only concerned the architecture; a good example was the time delay neural network architecture, also called finite impulse response network.

Standard supervised learning (SL), on the one hand, views time series prediction essentially as nonlinear regression; the fact that we are dealing with a time series is basically ignored. Temporal difference (TD) learning, on the other hand, takes a different approach: it adjusts the parameters based on differences between successive predictions in time (Sutton, 1988). TD learning has been shown to be very successful in the context of games such as backgammon (Tesauro, 1992). This paper investigates whether the TD paradigm can also be applied to the somewhat different task of time series prediction.

This paper is organized as follows: after briefly reviewing TD learning, Section 2 focuses on the application of TD learning to multi-step prediction of time series, and contrasts it to supervised learning (see Figure 1). Section 3 then describes the architectures and cost function as well as the data set used in the experiments. Section 4 presents the results, and Section 5 summarizes the paper.

# 2  TD learning for nonlinear, direct multi-step predictors

The key idea behind TD learning is that errors (used for gradient descent) are based on predictions that are adjacent in time. This is in contrast to the traditional SL approach where the errors are based on the difference between the prediction and the observed value. The general expression for the TD weight update rule (linear case), TD($\lambda$), is given by (Sutton, 88)

$$\Delta w_t = \eta \left( \hat{y}_{t+1} - \hat{y}_t \right) \sum_{k=1}^{t} \lambda^{t-k} \nabla_w \hat{y}_k \qquad (1)$$

where $\eta$ is the learning rate; $\hat{y}_{t+1}$ and $\hat{y}_t$ are two adjacent predictions of the *equivalent* target; $\lambda$ is the recency weight with $0 \le \lambda \le 1$; and $\nabla_w \hat{y}_k$ is the gradient of the prediction at time $k$ with respect to the weights of the network.

In Equation (1), we use the present weights to calculate the predictions $\hat{y}_t$ and $\hat{y}_{t+1}$ and we use the past weights to calculate the past gradients. In our experiments, since $\hat{y}_t$ is an output of a nonlinear connectionist network, we form $\hat{y}_t$ by propagating it through a multilayer network with hidden units and we backpropagate weight changes by applying the chain rule to the gradient $\nabla_w \hat{y}_k$ with respect to the hidden layer activation function. Several variants of TD($\lambda$) exist: TD(0) only forms gradients based on the present pair of predictions $(\hat{y}_t, \hat{y}_{t+1})$; TD(1) continually adds gradients with no weighting of recency; and in the general case, TD($\lambda$) weights the $k$th past gradient by a recency weight of $\lambda^k$. As will be shown in the subsequent section, in our example, TD($\lambda$) tends to lead to the best results for $\lambda$ around 0.3 (the optimal value of $\lambda$ cannot be determined by first principles).

In multi-step prediction, we *directly* predict the value of a time series $n$ time steps into the future ($y_{t+n}$ denotes the observed value at $t + n$), given a set of $m$ past values at time $t$ denoted by the observation vector $\mathbf{x}_t : (y_t, y_{t-1}, \ldots y_{t-m-1})$. To cast the multi-step prediction problem into the TD framework, we first form an overlapping sequence of predictions as described by Sutton (1988): For an $n$-step ahead prediction problem, we form $n$ successive predictions of the same target $y_{t+n}$: $\hat{y}_t^n, \hat{y}_{t+1}^{n-1}, \ldots \hat{y}_{t+n-1}^1$ ($\hat{y}_t^\delta$ is the prediction at time $t$ for the time series $\delta$ steps ahead). At each time step, we form two sets of predictions $\hat{y}_t^\delta$ and $\hat{y}_{t+1}^{\delta-1}$ based on the observation pair $(\mathbf{x}_t, \mathbf{x}_{t+1})$. The corresponding weight update at time $t$ involves the temporal difference of the these equivalent predictions:

$$\Delta w_t^\delta = \eta \left( \hat{y}_{t+1}^{\delta-1} - \hat{y}_t^\delta \right) \sum_{k=1}^{t} \lambda^{t-k} \nabla_w \hat{y}_k^\delta \tag{2}$$

Equation (2) shows that the TD algorithm reduces to the SL algorithm for single-step predictions, since there is no temporal structure revealed in time (i.e. the actual value is available with the first observation pair $(\mathbf{x}_t, \mathbf{x}_{t+1})$ at time $t$, and therefore $\hat{y}_{t+1}^0 = y_{t+1}$). However, for a multi-step prediction problem, temporal structure exists in the revelation of the observation vectors. on-line, the

To differentiate the two algorithms, Figure 1 depicts the backpropagation of errors using SL and TD learning algorithms. In SL, errors are generated by the squared difference between predicted and target values: network training simply tries to minimize the error function based on *structural* difference between the predicted ($\hat{y}_t^k$, or simply $\hat{y}_k$) and target values ($y_k$), see (Figure 1(a)). In (Figure 1(b)), TD learning minimizes a different error function—the difference between successive predictions $\hat{y}_k(t)$ and $\hat{y}_{k-1}(t + 1)$. Note, in the case of a noiseless time series, we do not expect to see a difference in performance between SL and TD learning, as the actual values of the time series are accurate descriptors of the system's output. In the case of a noisy time series, we conjecture that TD learning provides a better teaching signal than simply using the noisy observable. In this paper, we begin by comparing the performance of SL and TD learning on a low noise deterministic time series.

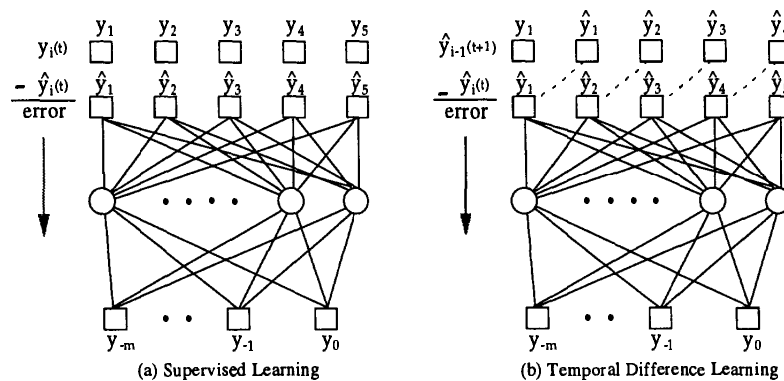

(a) Supervised Learning         (b) Temporal Difference Learning

Figure 1: Backpropagation for Supervised and Temporal Difference learning.

## 3   Architecture and Data

The multi-step prediction problem we chose is to *directly* predict the next five values of a time series given the past five values. In comparing the two algorithms, we examine two architectures and compare their performance on a real-world dataset.

**Architecture.** Two network architectures were chosen to compare the SL and TD algorithms in the multi-step prediction task.

- **Separate hidden units.** The first architecture consists of five separate prediction networks, each forming a single prediction $\hat{y}_i$ for the $i$th step-ahead prediction of the series. The five outputs correspond to the predictions $\hat{y}_1$ through $\hat{y}_5$. Each network has five input units (corresponding to the past five values of the time series), 10 hidden units (arbitrarily chosen) and a single linear output for each task.

- **Shared hidden units.** The second architecture is a single network with five outputs corresponding to the five predictions $(\hat{y}_1 \dots \hat{y}_5)$. The network has the same five inputs, but has 20 tanh hidden units.

**Cost Function.** We train on sum squared error, weighting all five predictions equally. In the supervised learning case, the predictions are compared to the actual values, while in the TD case, errors are calculated based on successive predictions:

$$E_{TD}(t) = \sum_{k=1}^{5}(\hat{y}_{k-1}(t+1) - \hat{y}_k(t))^2 \tag{3}$$

**Search.** In network training, we use batch updates, i.e., update weights after each pass through the training data. Network training continues until the error on the cross-validation set stagnates or begins to increase. For networks trained by TD learning, $\lambda$, the recency weight, ranges from $0 \leq \lambda \leq 0.5$.

**Data.** We use the laser data from the *Santa Fe Competition*.[1] The data are intensity measurements of a $NH_3$ laser in a chaotic state, exhibiting Lorenz-like dynamics. We use the 1,000 competition data points for training, but also 1,000 further points for cross-validation of our model and 2,000 further points for testing. We depart from the competition rules in order to get higher statistical significance.

## 4   Results

**Learning curves.** We begin our analysis by plotting the squared error normalized by the variance for each of the five output units $(\hat{y}_1 - \hat{y}_5)$ as a function of training time for both

learning algorithms in Figure 2.[2] In the SL case, although the sum of the five curves monotonically decreases, they individually can fall and rise again. This plot shows the trade-offs in multi-task learning. For example, $\hat{y}_3$ is learned early and then levels off, because $\hat{y}_5$ is being learned. In the TD learning case, the five curves are ordered in the order that we would expect with the error associated with $\hat{y}_1$ always smaller than $\hat{y}_2$ and so on. This is expected, since the $i$th prediction $\hat{y}_i$ is driven by the prediction $\hat{y}_{i-1}$ projected one step into the future. We also note that the $\hat{y}_1$ curve is similar for both paradigms, since the error is driven by the same observed value $y_1$.

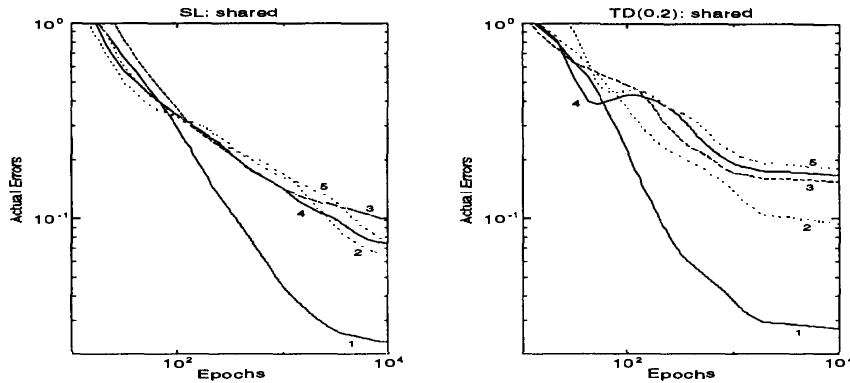

Figure 2: $E_{NMS}$ for each output versus training epochs (training set) for both supervised and TD learning. Typical runs are shown and both architectures exhibit the same behavior. Learning rates were varied during training to accelerate learning. In the TD(0.2) case, the leveling of the error curves at 2,000 epochs is due to a decrease in learning rate.

**Performance metric.** We compare the performances with the normalized mean square error,

$$E_{NMS} = \frac{\sum_{k=1}^{N}(y_k - \hat{y}_k)^2}{\sum_{k=1}^{N}(y_k - \text{mean})^2} \tag{4}$$

where N is the number of samples; $y_k$ and $\hat{y}_k$ are the actual and predicted values.

**Comparison between SL and TD learning on five-step prediction.** The longer the lead time for the forecast, the larger the expected difference in performance between TD and SL. We thus focus on five-step predictions where the difference between SL and TD is most pronounced from the set we considered ($\hat{y}_1$-$\hat{y}_5$). Figure 3 shows the individual performances of several runs for the task of direct five-step predictions. We vary the architecture (left side is shared hidden units, right side is separate hidden units), and we vary in each sub-plot the training (SL, TD(0), TD(0.2), TD(0.3), TD(0.5)). There is no large difference within TD for different values of $\lambda$. However, there is a significant difference between SL and TD: SL is better than TD. This result depends crucially on the fact that the data have very low noise (the main source of noise is just the quantization error of the 8-bit analog to digital converter).

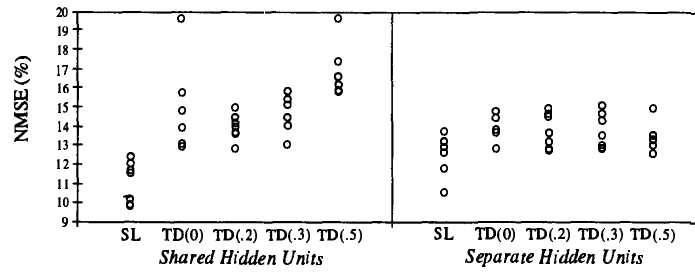

Figure 3: $E_{NMS}$ for the direct five-step prediction $\hat{y}_5$ for both architectures (test set).

**Performance comparison between single task and multi-task SL learning.** Still for the task of five-step ahead prediction, we wanted to investigate whether predicting several tasks versus predicting a single task is beneficial. Comparing the SL column between the left and the right side of Figure 3 shows no significant difference. To eliminate the hypotheses that the performance was limited by the available number of hidden units, we also ran networks with only a single output unit for the one task of $\hat{y}_5$ and allocated up to 50 hidden units. The performance still remains the same. Thus, the fact that additional tasks *did not hurt* the performance indicates that the networks had sufficient resources. The fact that additional tasks *did not help* the performance indicates that there is only little noise in the data. This is different in high-noise problems, e.g., Weigend, Huberman and Rumelhart (1992) used multi-task predictions for currency exchange rates. See also Breiman and Friedman (1994), Caruana (1994), and Nix and Weigend (1995) for further discussions on multi-task learning.

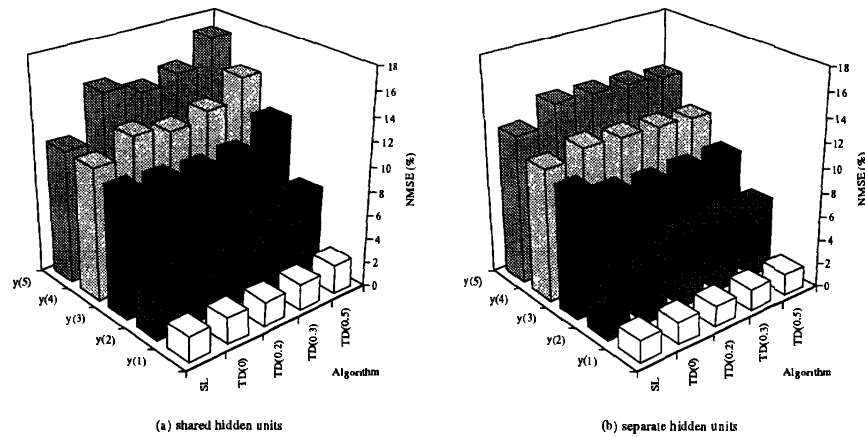

(a) shared hidden units                                      (b) separate hidden units

Figure 4: Summary of test set performance for SL and $TD(\lambda)$ for both architectures.

**SL vs. TD learning.** Figure 4 and Table 1 summarize the performance of all the networks. As stated earlier, networks trained by SL outperformed networks trained by $TD(\lambda)$ for $\hat{y}_5$. For earlier predictions $\hat{y}_1 - \hat{y}_4$, no significant performance discrepancies exist for either architecture or learning algorithm. Note, for $\hat{y}_1$, the results for the TD(0) and SL networks are equivalent for the separate hidden unit network, since the error function was equivalent for both algorithms. Among the networks trained by TD learning, the TD(0.3) and TD(0.5) networks exhibit the best average performance.

| Prediction | SL | TD(0) | TD(0.2) | TD(0.3) | TD(0.5) | ± σ |
|---|---|---|---|---|---|---|
| *Shared:* | | | | | | |
| $\hat{y}_1$ | 2.1 | 2.1 | 2.0 | 2.1 | 2.3 | 0.13 |
| $\hat{y}_2$ | 7.0 | 7.1 | 6.6 | 6.7 | 7.0 | 0.63 |
| $\hat{y}_3$ | 10.2 | 10.0 | 9.8 | 10.0 | 11.8 | 0.83 |
| $\hat{y}_4$ | 10.9 | 12.4 | 11.8 | 12.5 | 14.4 | 0.91 |
| $\hat{y}_5$ | 11.0 | 14.8 | 13.9 | 14.6 | 16.8 | 1.30 |
| *Separate:* | | | | | | |
| $\hat{y}_1$ | 1.8 | 1.8 | 1.7 | 1.7 | 1.7 | 0.05 |
| $\hat{y}_2$ | 6.2 | 5.8 | 5.8 | 6.0 | 6.1 | 0.25 |
| $\hat{y}_3$ | 10.2 | 9.0 | 8.7 | 8.8 | 8.9 | 0.72 |
| $\hat{y}_4$ | 10.8 | 11.5 | 11.3 | 11.2 | 11.0 | 0.57 |
| $\hat{y}_5$ | 12.2 | 13.9 | 13.8 | 13.7 | 13.4 | 0.85 |

Table 1: Summary of test set performance for SL and TD(λ) ($E_{NMS}$ is given in percent and σ is the empirical standard deviation averaged over each row).

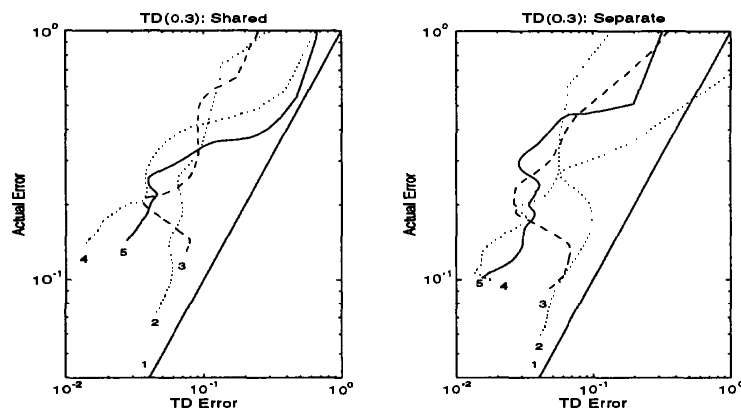

Figure 5: TD errors versus actual errors for both architectures (λ = 0.3)

**Actual versus Temporal Difference errors.** Since the temporal difference learning rule is based on the error between neighboring predictions in time, the following question may arise: How do the actual errors (between the predicted and observed values) vary with respect to the TD errors (between adjacent predictions of the same target) during training? Figure 5 plots the actual errors versus TD errors for λ=0.3 for both architectures. For both architectures, the TD errors are smaller than the actual errors. From Figure 5, in both architectures, nearer prediction errors influence further predictions. For noisier data sets, the curves in Figure 5, we expect, will take on an upward slope at the end of training, signaling that overfitting has begun.

## 5 Conclusions

We explored the application of Temporal Difference (TD) learning to forecasting real-valued time series, as opposed to game-like situations. After relating TD learning to supervised

learning (SL) from a general perspective, we compare and analyze the performance of both paradigms on a specific data set, the deterministic chaotic laser data used in the Santa Fe Competition. For this low-noise time series data we find

- Within each paradigm, the learning curves (for the individual outputs) do not depend on the specific architecture of shared or separate hidden units.

- Across the two paradigms, the learning curves in SL show a larger error trade-off amongst the individual outputs than in TD learning.

- For the longest lead time considered (five-step ahead predictions), the difference between SL and TD is most pronounced: SL outperforms TD.

- Within SL, giving the network additional tasks (such as not only predicting the five-step ahead forecast but also the intermediate steps and using shared hidden units) did not change the performance compared to a single output with separate hidden units.

- The best choice of the recency weight $\lambda$ appears to be in the range of $0.2 \leq \lambda \leq 0.5$.

- Plotting the TD error versus the actual error is a useful new diagnostic, particularly on out-of-sample data for noisy problems.

The performance of TD learning in comparison to SL depends on the amount of noise present in the data. For the low noise time series used in this paper, there is no advantage in using TD learning over SL. At present, we are comparing the two paradigms on noisy real-world data where overfitting is a serious challenge.

## Acknowledgments

We thank Richard Sutton for his suggestions concerning the implementation. Andreas Weigend acknowledges support from the National Science Foundation, Research Initiation Grant No. RIA ECS-9309786.

## Footnotes

[1]The data set and several predictions and characterizations are described in the volume edited by Weigend and Gershenfeld (1994). The data is available by anonymous ftp at `ftp.cs.colorado.edu` in `/pub/Time-Series/SantaFe` as `A.dat`. See also `http://www.cs.colorado.edu/Time-Series/TSWelcome.html` for further analyses of this and other time series data sets.

[2]The normalized mean square errors $E_{NMS}$ do not start with 1.0 as it would have been the case for very small initial weights; we ran the experiments (for no particular reason) with rather large initial weights, drawn from a uniform distribution between -1 and +1.

## References

L. Breiman and J.H. Friedman (1994) "A New Look at Multiple Outputs." Abstract, Neural Networks for Computing, Snowbird, UT, April 1994

R.A. Caruana (1994) "Multitask Connectionist Learning." In *Proceedings of the 1993 Connectionist Models Summer School*, edited by M. C. Mozer, P. Smolensky, D. S. Touretzky, J. L. Elman, and A. S. Weigend, p. 372-379. Hillsdale, NJ: Erlbaum Associates.

D.A. Nix and A.S. Weigend (1995) "Learning Local Error Bars for Nonlinear Regression." In *Advances in Neural Information Processing Systems 7 (NIPS*94, this volume)*. San Francisco, CA: Morgan Kaufmann.

R.S. Sutton (1988) "Learning to Predict by the Methods of Temporal Differences." *Machine Learning* **3**: 9-44.

G. Tesauro (1992) "Practical Issues in Temporal Difference learning." *Machine Learning* **8**: 257-277.

A.S. Weigend & N.A. Gershenfeld, eds. (1994) *Time Series Prediction: Forecasting the Future and Understanding the Past*. Reading, MA: Addison-Wesley.

A.S. Weigend, B.A. Huberman, and D.E. Rumelhart (1992) "Predicting Sunspots and Exchange Rates with Connectionist Networks." In *Nonlinear Modeling and Forecasting*, edited by M. Casdagli, and S. Eubank, p. 395-432. Redwood City, CA: Addison-Wesley.
